# Recovery of Jointly Sparse Signals from Few Random Projections

**Michael B. Wakin**
ECE Department
Rice University
wakin@rice.edu

**Marco F. Duarte**
ECE Department
Rice University
duarte@rice.edu

**Shriram Sarvotham**
ECE Department
Rice University
shri@rice.edu

**Dror Baron**
ECE Department
Rice University
drorb@rice.edu

**Richard G. Baraniuk**
ECE Department
Rice University
richb@rice.edu

## Abstract

Compressed sensing is an emerging field based on the revelation that a small group of linear projections of a sparse signal contains enough information for reconstruction. In this paper we introduce a new theory for *distributed compressed sensing* (DCS) that enables new distributed coding algorithms for multi-signal ensembles that exploit both intra- and inter-signal correlation structures. The DCS theory rests on a new concept that we term the *joint sparsity* of a signal ensemble. We study three simple models for jointly sparse signals, propose algorithms for joint recovery of multiple signals from incoherent projections, and characterize theoretically and empirically the number of measurements per sensor required for accurate reconstruction. In some sense DCS is a framework for distributed compression of sources with memory, which has remained a challenging problem in information theory for some time. DCS is immediately applicable to a range of problems in sensor networks and arrays.

## 1 Introduction

*Distributed communication, sensing, and computing* [13, 17] are emerging fields with numerous promising applications. In a typical setup, large groups of cheap and individually unreliable nodes may collaborate to perform a variety of data processing tasks such as sensing, data collection, classification, modeling, tracking, and so on. As individual nodes in such a network are often battery-operated, power consumption is a limiting factor, and the reduction of communication costs is crucial. In such a setting, *distributed source coding* [8, 13, 14, 17] may allow the sensors to save on communication costs. In the Slepian-Wolf framework for lossless distributed coding [8, 14], the availability of *correlated side information* at the decoder enables the source encoder to communicate losslessly at the conditional entropy rate, rather than the individual entropy. Because sensor networks and arrays rely on data that often exhibit strong spatial correlations [13, 17], distributed compression can reduce the communication costs substantially, thus enhancing battery life. Unfortunately, distributed compression schemes for sources with memory are not yet mature [8, 13, 14, 17].

We propose a new approach for distributed coding of correlated sources whose signal correlations take the form of a sparse structure. Our approach is based on another emerging field known as *compressed sensing* (CS) [4, 9]. CS builds upon the groundbreaking work of Candès et al. [4] and Donoho [9], who showed that signals that are *sparse* relative to a known basis can be recovered from a small number of nonadaptive linear projections onto a second basis that is incoherent with the first. (A random basis provides such incoherence with high probability. Hence CS with random projections is *universal* — the signals can be reconstructed if they are sparse relative to *any* known basis.) The implications of CS for signal acquisition and compression are very promising. With no a priori knowledge of a signal's structure, a sensor node could simultaneously acquire and compress that signal, preserving the critical information that is extracted only later at a fusion center.

In our framework for *distributed compressed sensing* (DCS), this advantage is particularly compelling. In a typical DCS scenario, a number of sensors measure signals that are each individually sparse in some basis and also correlated from sensor to sensor. Each sensor *independently* encodes its signal by projecting it onto another, incoherent basis (such as a random one) and then transmits just a few of the resulting coefficients to a single collection point. Under the right conditions, a decoder at the collection point can reconstruct each of the signals precisely. The DCS theory rests on a concept that we term the *joint sparsity* of a signal ensemble. We study in detail three simple models for jointly sparse signals, propose tractable algorithms for joint recovery of signal ensembles from incoherent projections, and characterize theoretically and empirically the number of measurements per sensor required for reconstruction. While the sensors operate entirely without collaboration, joint decoding can recover signals using far fewer measurements per sensor than would be required for separable CS recovery. This paper presents our specific results for one of the three models; the other two are highlighted in our papers [1, 2, 11].

## 2   Sparse Signal Recovery from Incoherent Projections

In the traditional CS setting, we consider a single signal $x \in \mathbb{R}^N$, which we assume to be sparse in a known orthonormal basis or frame $\Psi = [\psi_1, \psi_2, \ldots, \psi_N]$. That is, $x = \Psi\theta$ for some $\theta$, where $\|\theta\|_0 = K$ holds.[1] The signal $x$ is observed indirectly via an $M \times N$ measurement matrix $\Phi$, where $M < N$. We let $y = \Phi x$ be the observation vector, consisting of the $M$ inner products of the measurement vectors against the signal. The $M$ rows of $\Phi$ are the measurement vectors, against which the signal is projected. These rows are chosen to be *incoherent* with $\Psi$ — that is, they each have non-sparse expansions in the basis $\Psi$ [4, 9]. In general, $\Phi$ meets the necessary criteria when its entries are drawn *randomly*, for example independent and identically distributed (i.i.d.) Gaussian.

Although the equation $y = \Phi x$ is underdetermined, it is possible to recover $x$ from $y$ under certain conditions. In general, due to the incoherence between $\Phi$ and $\Psi$, $\theta$ can be recovered by solving the $\ell_0$ optimization problem

$$\widehat{\theta} = \arg\min \|\theta\|_0 \quad \text{s.t. } y = \Phi\Psi\theta.$$

In principle, remarkably few random measurements are required to recover a $K$-sparse signal via $\ell_0$ minimization. Clearly, more than $K$ measurements must be taken to avoid ambiguity; in theory, $K + 1$ random measurements will suffice [2]. Unfortunately, solving this $\ell_0$ optimization problem appears to be NP-hard [6], requiring a combinatorial enumeration of the $\binom{N}{K}$ possible sparse subspaces for $\theta$.

The amazing revelation that supports the CS theory is that a much simpler problem yields an equivalent solution (thanks again to the incoherence of the bases): we need only solve

for the $\ell_1$-sparsest vector $\theta$ that agrees with the observed coefficients $y$ [4, 9]

$$\widehat{\theta} = \arg\min \|\theta\|_1 \quad \text{s.t. } y = \Phi\Psi\theta.$$

This optimization problem, known also as Basis Pursuit (BP) [7], is significantly more tractable and can be solved with traditional linear programming techniques. There is no free lunch, however; more than $K + 1$ measurements will be required in order to recover sparse signals. In general, there exists a constant oversampling factor $c = c(K, N)$ such that $cK$ measurements suffice to recover $x$ with very high probability [4, 9]. Commonly quoted as $c = O(\log(N))$, we have found that $c \approx \log_2(1 + N/K)$ provides a useful rule-of-thumb [2]. At the expense of slightly more measurements, greedy algorithms have also been developed to recover $x$ from $y$. One example, known as Orthogonal Matching Pursuit (OMP) [15], requires $c \approx 2\ln(N)$. We exploit both BP and greedy algorithms for recovering jointly sparse signals.

## 3   Joint Sparsity Models

In this section, we generalize the notion of a signal being sparse in some basis to the notion of an ensemble of signals being *jointly sparse*. We consider three different *joint sparsity models* (JSMs) that apply in different situations. In most cases, each signal is itself sparse, and so we could use the CS framework from above to encode and decode each one separately. However, there also exists a framework wherein a *joint representation* for the ensemble uses fewer total vectors.

We use the following notation for our signal ensembles and measurement model. Denote the *signals* in the ensemble by $x_j$, $j \in \{1, 2, \ldots, J\}$, and assume that each signal $x_j \in \mathbb{R}^N$. We assume that there exists a known *sparse basis* $\Psi$ for $\mathbb{R}^N$ in which the $x_j$ can be sparsely represented. Denote by $\Phi_j$ the *measurement matrix* for signal $j$; $\Phi_j$ is $M_j \times N$ and, in general, the entries of $\Phi_j$ are different for each $j$. Thus, $y_j = \Phi_j x_j$ consists of $M_j < N$ *incoherent measurements* of $x_j$.

**JSM-1: Sparse common component + innovations.** In this model, all signals share a *common* sparse component while each individual signal contains a sparse *innovation* component; that is,

$$x_j = z_C + z_j, \quad j \in \{1, 2, \ldots, J\}$$

with

$$z_C = \Psi\theta_C, \ \|\theta_C\|_0 = K \qquad \text{and} \qquad z_j = \Psi\theta_j, \ \|\theta_j\|_0 = K_j.$$

Thus, the signal $z_C$ is common to all of the $x_j$ and has sparsity $K$ in basis $\Psi$. The signals $z_j$ are the unique portions of the $x_j$ and have sparsity $K_j$ in the same basis. A practical situation well-modeled by JSM-1 is a group of sensors measuring temperatures at a number of outdoor locations throughout the day. The temperature readings $x_j$ have both temporal (intra-signal) and spatial (inter-signal) correlations. Global factors, such as the sun and prevailing winds, could have an effect $z_C$ that is both common to all sensors and structured enough to permit sparse representation. More local factors, such as shade, water, or animals, could contribute localized innovations $z_j$ that are also structured (and hence sparse). Similar scenarios could be imagined for a network of sensors recording other phenomena that change smoothly in time and in space and thus are highly correlated.

**JSM-2: Common sparse supports.** In this model, all signals are constructed from the same sparse set of basis vectors, but with different coefficients; that is,

$$x_j = \Psi\theta_j, \quad j \in \{1, 2, \ldots, J\},$$

where each $\theta_j$ is supported only on the same $\Omega \subset \{1, 2, \ldots, N\}$ with $|\Omega| = K$. Hence, all signals have $\ell_0$ sparsity of $K$, and all are constructed from the same $K$ basis elements, but with arbitrarily different coefficients. A practical situation well-modeled by JSM-2 is where multiple sensors acquire the same signal but with phase shifts and attenuations

caused by signal propagation. In many cases it is critical to recover each one of the sensed signals, such as in many acoustic localization and array processing algorithms. Another useful application for JSM-2 is MIMO communication [16].

**JSM-3: Nonsparse common + sparse innovations.** This model extends JSM-1 so that the common component need no longer be sparse in any basis; that is,

$$x_j = z_C + z_j, \quad j \in \{1, 2, \ldots, J\}$$

with

$$z_C = \Psi\theta_C \qquad \text{and} \qquad z_j = \Psi\theta_j, \ \ \|\theta_j\|_0 = K_j,$$

but $z_C$ is not necessarily sparse in the basis $\Psi$. We also consider the case where the supports of the innovations are shared for all signals, which extends JSM-2. A practical situation well-modeled by JSM-3 is where several sources are recorded by different sensors together with a background signal that is not sparse in any basis. Consider, for example, a computer vision-based verification system in a device production plant. Cameras acquire snapshots of components in the production line; a computer system then checks for failures in the devices for quality control purposes. While each image could be extremely complicated, the ensemble of images will be highly correlated, since each camera is observing the same device with minor (sparse) variations. JSM-3 could also be useful in some non-distributed scenarios. For example, it motivates the compression of data such as video, where the innovations or differences between video frames may be sparse, even though a single frame may not be very sparse. In general, JSM-3 may be invoked for ensembles with significant inter-signal correlations but insignificant intra-signal correlations.

## 4 Recovery of Jointly Sparse Signals

In a setting where a network or array of sensors may encounter a collection of jointly sparse signals, and where a centralized reconstruction algorithm is feasible, the number of incoherent measurements required by each sensor can be reduced. For each JSM, we propose algorithms for joint signal recovery from incoherent projections and characterize theoretically and empirically the number of measurements per sensor required for accurate reconstruction. We focus in particular on JSM-3 in this paper but also overview our results for JSMs 1 and 2, which are discussed in further detail in our papers [1, 2, 11].

### 4.1 JSM-1: Sparse common component + innovations

For this model (see also [1, 2]), we have proposed an analytical framework inspired by the principles of information theory. This allows us to characterize the measurement rates $M_j$ required to *jointly* reconstruct the signals $x_j$. The measurement rates relate directly to the signals' *conditional sparsities*, in parallel with the Slepian-Wolf theory. More specifically, we have formalized the following intuition. Consider the simple case of $J = 2$ signals. By employing the CS machinery, we might expect that (*i*) $(K + K_1)c$ coefficients suffice to reconstruct $x_1$, (*ii*) $(K + K_2)c$ coefficients suffice to reconstruct $x_2$, yet only (*iii*) $(K + K_1 + K_2)c$ coefficients should suffice to reconstruct both $x_1$ and $x_2$, since we have $K + K_1 + K_2$ nonzero elements in $x_1$ and $x_2$. In addition, given the $(K + K_1)c$ measurements for $x_1$ as side information, and assuming that the partitioning of $x_1$ into $z_C$ and $z_1$ is known, $cK_2$ measurements that describe $z_2$ should allow reconstruction of $x_2$. Formalizing these arguments allows us to establish theoretical lower bounds on the required measurement rates at each sensor; Fig.1(a) shows such a bound for the case of $J = 2$ signals.

We have also established upper bounds on the required measurement rates $M_j$ by proposing a specific algorithm for reconstruction [1]. The algorithm uses carefully designed measurement matrices $\Phi_j$ (in which some rows are identical and some differ) so that the resulting measurements can be combined to allow step-by-step recovery of the sparse components. The theoretical rates $M_j$ are below those required for separable CS recovery of each signal $x_j$ (see Fig. 1(a)). We also proposed a reconstruction technique based on a single execution of a linear program, which seeks the sparsest components $[z_C; \ z_1; \ \ldots \ z_J]$ that

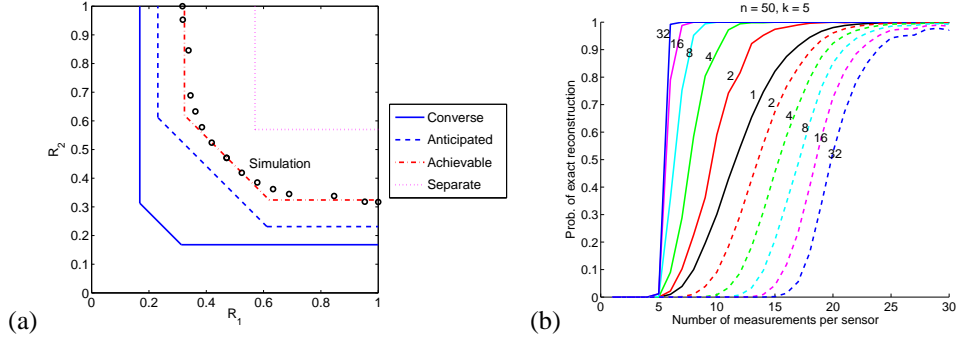

Figure 1: *(a) Converse bounds and achievable measurement rates for $J = 2$ signals with common sparse component and sparse innovations (JSM-1). We fix signal lengths $N = 1000$ and sparsities $K = 200$, $K_1 = K_2 = 50$. The measurement rates $R_j := M_j/N$ reflect the number of measurements normalized by the signal length. Blue curves indicate our theoretical and anticipated converse bounds; red indicates a provably achievable region, and pink denotes the rates required for separable CS signal reconstruction. (b) Reconstructing a signal ensemble with common sparse supports (JSM-2). We plot the probability of perfect reconstruction via DCS-SOMP (solid lines) and independent CS reconstruction (dashed lines) as a function of the number of measurements per signal $M$ and the number of signals $J$. We fix the signal length to $N = 50$ and the sparsity to $K = 5$. An oracle encoder that knows the positions of the large coefficients would use 5 measurements per signal.*

account for the observed measurements. Numerical simulations support such an approach (see Fig.1(a)). Future work will extend JSM-1 to $\ell_p$-compressible signals, $0 < p \le 1$.

### 4.2 JSM-2: Common sparse supports

Under the JSM-2 signal ensemble model (see also [2, 11]), independent recovery of each signal via $\ell_1$ minimization would require $cK$ measurements per signal. However, algorithms inspired by conventional greedy pursuit algorithms (such as OMP [15]) can substantially reduce this number. In the single-signal case, OMP iteratively constructs the sparse support set $\Omega$; decisions are based on inner products between the columns of $\Phi\Psi$ and a residual. In the multi-signal case, there are more clues available for determining the elements of $\Omega$.

To establish a theoretical justification for our approach, we first proposed a simple One-Step Greedy Algorithm (OSGA) [11] that combines all of the measurements and seeks the largest correlations with the columns of the $\Phi_j\Psi$. We established that, assuming that $\Phi_j$ has i.i.d. Gaussian entries and that the nonzero coefficients in the $\theta_j$ are i.i.d. Gaussian, then with $M \ge 1$ measurements per signal, OSGA recovers $\Omega$ with probability approaching 1 as $J \to \infty$. Moreover, with $M \ge K$ measurements per signal, OSGA recovers all $x_j$ with probability approaching 1 as $J \to \infty$. This meets the theoretical lower bound for $M_j$.

In practice, OSGA can be improved using an iterative greedy algorithm. We proposed a simple variant of Simultaneous Orthogonal Matching Pursuit (SOMP) [16] that we term DCS-SOMP [11]. For this algorithm, Fig. 1(b) plots the performance as the number of sensors varies from $J = 1$ to 32. We fix the signal lengths at $N = 50$ and the sparsity of each signal to $K = 5$. With DCS-SOMP, for perfect reconstruction of all signals the average number of measurements per signal decreases as a function of $J$. The trend suggests that, for very large $J$, close to $K$ measurements per signal should suffice. On the contrary, with independent CS reconstruction, for perfect reconstruction of all signals the number of measurements per sensor *increases* as a function of $J$. This surprise is due to the fact that each signal will experience an independent probability $p \le 1$ of successful reconstruction; therefore the overall probability of complete success is $p^J$. Consequently, each sensor must compensate by making additional measurements.

### 4.3 JSM-3: Nonsparse common + sparse innovations

The JSM-3 signal ensemble model provides a particularly compelling motivation for joint recovery. Under this model, no individual signal $x_j$ is sparse, and so separate signal recovery would require fully $N$ measurements per signal. As in the other JSMs, however, the commonality among the signals makes it possible to substantially reduce this number.

Our recovery algorithms are based on the observation that if the common component $z_C$ were known, then each innovation $z_j$ could be estimated using the standard single-signal CS machinery on the adjusted measurements $y_j - \Phi_j z_C = \Phi_j z_j$. While $z_C$ is not known in advance, it can be *estimated* from the measurements. In fact, across all $J$ sensors, a total of $\sum_j M_j$ random projections of $z_C$ are observed (each corrupted by a contribution from one of the $z_j$). Since $z_C$ is not sparse, it cannot be recovered via CS techniques, but when the number of measurements is sufficiently large ($\sum_j M_j \gg N$), $z_C$ can be estimated using standard tools from linear algebra. A key requirement for such a method to succeed in recovering $z_C$ is that each $\Phi_j$ be different, so that their rows combine to span all of $\mathbb{R}^N$. In the limit, $z_C$ can be recovered while still allowing each sensor to operate at the minimum measurement rate dictated by the $\{z_j\}$. A prototype algorithm, which we name Transpose Estimation of Common Component (TECC), is listed below, where we assume that each measurement matrix $\Phi_j$ has i.i.d. $\mathcal{N}(0, \sigma_j^2)$ entries.

#### TECC Algorithm for JSM-3

1. **Estimate common component:** Define the matrix $\widehat{\Phi}$ as the concatenation of the regularized individual measurement matrices $\widehat{\Phi}_j = \frac{1}{M_j \sigma_j^2} \Phi_j$, that is, $\widehat{\Phi} = [\widehat{\Phi}_1, \widehat{\Phi}_2, \dots, \widehat{\Phi}_J]$.
   Calculate the estimate of the common component as $\widehat{z_C} = \frac{1}{J} \widehat{\Phi}^T y$.

2. **Estimate measurements generated by innovations:** Using the previous estimate, subtract the contribution of the common part on the measurements and generate estimates for the measurements caused by the innovations for each signal: $\widehat{y}_j = y_j - \Phi_j \widehat{z_C}$.

3. **Reconstruct innovations:** Using a standard single-signal CS reconstruction algorithm, obtain estimates of the innovations $\widehat{z}_j$ from the estimated innovation measurements $\widehat{y}_j$.

4. **Obtain signal estimates:** Sum the above estimates, letting $\widehat{x}_j = \widehat{z_C} + \widehat{z}_j$.

The following theorem shows that asymptotically, by using the TECC algorithm, each sensor need only measure at the rate dictated by the sparsity $K_j$.

**Theorem 1** [2] *Assume that the nonzero expansion coefficients of the sparse innovations $z_j$ are i.i.d. Gaussian random variables and that their locations are uniformly distributed on $\{1, 2, ..., N\}$. Then the following statements hold:*

1. *Let the measurement matrices $\Phi_j$ contain i.i.d. $\mathcal{N}(0, \sigma_j^2)$ entries with $M_j \geq K_j + 1$. Then each signal $x_j$ can be recovered using the TECC algorithm with probability approaching 1 as $J \to \infty$.*

2. *Let $\Phi_j$ be a measurement matrix with $M_j \leq K_j$ for some $j \in \{1, 2, ..., J\}$. Then with probability 1, the signal $x_j$ cannot be uniquely recovered by any algorithm for any $J$.*

For large $J$, the measurement rates permitted by Statement 1 are the lowest possible for *any* reconstruction strategy on JSM-3 signals, even neglecting the presence of the nonsparse component. Thus, Theorem 1 provides a tight achievable and converse for JSM-3 signals. The CS technique employed in Theorem 1 involves combinatorial searches for estimating the innovation components. More efficient techniques could also be employed (including several proposed for CS in the presence of noise [3, 5, 7, 10, 12]).

While Theorem 1 suggests the theoretical gains from joint recovery as $J \to \infty$, practical gains can also be realized with a moderate number of sensors. For example, suppose in the TECC algorithm that the initial estimate $\widehat{z_C}$ is not accurate enough to enable correct

identification of the sparse innovation supports $\{\Omega_j\}$. In such a case, it may still be possible for a rough approximation of the innovations $\{z_j\}$ to help refine the estimate $\widehat{z_C}$. This in turn could help to refine the estimates of the innovations. Since each component helps to estimate the others, we propose an iterative algorithm for JSM-3 recovery. The Alternating Common and Innovation Estimation (ACIE) algorithm exploits the observation that once the basis vectors comprising the innovation $z_j$ have been identified in the index set $\Omega_j$, their effect on the measurements $y_j$ can be removed to aid in estimating $z_C$.

### ACIE Algorithm for JSM-3

1. **Initialize:** Set $\widehat{\Omega}_j = \emptyset$ for each $j$. Set the iteration counter $\ell = 1$.

2. **Estimate common component:** Let $\Phi_{j,\widehat{\Omega}_j}$ be the $M_j \times |\widehat{\Omega}_j|$ submatrix obtained by sampling the columns $\widehat{\Omega}_j$ from $\Phi_j$ and construct an $M_j \times (M_j - |\widehat{\Omega}_j|)$ matrix $Q_j = [q_{j,1} \ \ldots \ q_{j,M_j-|\widehat{\Omega}_j|}]$ having orthonormal columns that span the orthogonal complement of $\mathrm{colspan}(\Phi_{j,\widehat{\Omega}_j})$. Remove the projection of the measurements into the aforementioned span to obtain measurements caused exclusively by vectors not in $\widehat{\Omega}_j$, letting $\widetilde{y}_j = Q_j^T y_j$ and $\widetilde{\Phi}_j = Q_j^T \Phi_j$. Use the modified measurements $\widetilde{Y} = \left[\widetilde{y}_1^T \ \widetilde{y}_2^T \ \ldots \ \widetilde{y}_J^T\right]^T$ and modified holographic basis $\widetilde{\Phi} = \left[\widetilde{\Phi}_1^T \ \widetilde{\Phi}_2^T \ \ldots \ \widetilde{\Phi}_J^T\right]^T$ to refine the estimate of the measurements caused by the common part of the signal, setting $\widetilde{z_C} = \widetilde{\Phi}^\dagger \widetilde{Y}$, where $A^\dagger = (A^T A)^{-1} A^T$ denotes the pseudoinverse of matrix $A$.

3. **Estimate innovation supports:** For each signal $j$, subtract $\widetilde{z_C}$ from the measurements, $\widehat{y}_j = y_j - \Phi_j \widetilde{z_C}$, and estimate the sparse support of each innovation $\widehat{\Omega}_j$.

4. **Iterate:** If $\ell < L$, a preset number of iterations, then increment $\ell$ and return to Step 2. Otherwise proceed to Step 5.

5. **Estimate innovation coefficients:** For each signal $j$, estimate the coefficients for the indices in $\widehat{\Omega}_j$, setting $\widehat{\theta}_{j,\widehat{\Omega}_j} = \Phi_{j,\widehat{\Omega}_j}^\dagger (y_j - \Phi_j \widetilde{z_C})$, where $\widehat{\theta}_{j,\widehat{\Omega}_j}$ is a sampled version of the innovation's sparse coefficient vector estimate $\widehat{\theta}_j$.

6. **Reconstruct signals:** Estimate each signal as $\widehat{x}_j = \widetilde{z_C} + \widehat{z}_j = \widetilde{z_C} + \Phi_j \widehat{\theta}_j$.

In the case where the innovation support estimate is correct ($\widehat{\Omega}_j = \Omega_j$), the measurements $\widetilde{y}_j$ will describe only the common component $z_C$. If this is true for every signal $j$ and the number of remaining measurements $\sum_j M_j - KJ \geq N$, then $z_C$ can be perfectly recovered in Step 2. Because it may be difficult to correctly obtain all $\Omega_j$ in the first iteration, we find it preferable to run the algorithm for several iterations.

Fig. 2(a) shows that, for sufficiently large $J$, we can recover all of the signals with significantly fewer than $N$ measurements per signal. We note the following behavior in the graph. First, as $J$ grows, it becomes more difficult to perfectly reconstruct all $J$ signals. We believe this is inevitable, because even if $z_C$ were known without error, then perfect ensemble recovery would require the successful execution of $J$ *independent* runs of OMP. Second, for small $J$, the probability of success can decrease at high values of $M$. We believe this is due to the fact that initial errors in estimating $z_C$ may tend to be somewhat sparse (since $\widehat{z_C}$ roughly becomes an average of the signals $\{x_j\}$), and these sparse errors can mislead the subsequent OMP processes. For more moderate $M$, it seems that the errors in estimating $z_C$ (though greater) tend to be less sparse. We expect that a more sophisticated algorithm could alleviate such a problem, and we note that the problem is also mitigated at higher $J$.

Fig. 2(b) shows that when the sparse innovations share common supports we see an even greater savings. As a point of reference, a traditional approach to signal encoding would require 1600 total measurements to reconstruct these $J = 32$ nonsparse signals of length $N = 50$. Our approach requires only about 10 per sensor for a total of 320 measurements.

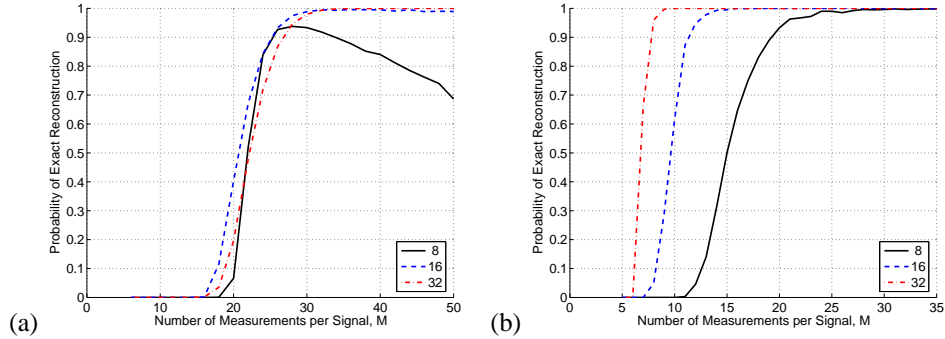

(a)                                                                (b)

Figure 2: *Reconstructing a signal ensemble with nonsparse common component and sparse innovations (JSM-3) using ACIE. (a) Reconstruction using OMP independently on each signal in Step 3 of the ACIE algorithm (innovations have arbitrary supports). (b) Reconstruction using DCS-SOMP jointly on all signals in Step 3 of the ACIE algorithm (innovations have identical supports). Signal length $N = 50$, sparsity $K = 5$. The common structure exploited by DCS-SOMP enables dramatic savings in the number of measurements. We average over 1000 simulation runs.*

**Acknowledgments**: Thanks to Emmanuel Candès, Hyeokho Choi, and Joel Tropp for informative and inspiring conversations.

## Footnotes

[1]The $\ell_0$ "norm" $\|\theta\|_0$ merely counts the number of nonzero entries in the vector $\theta$. CS theory also applies to signals for which $\|\theta\|_p \leq K$, where $0 < p \leq 1$; such extensions for DCS are a topic of ongoing research.

## References

[1] D. Baron, M. F. Duarte, S. Sarvotham, M. B. Wakin, and R. G. Baraniuk. An information-theoretic approach to distributed compressed sensing. In *Allerton Conf. Comm., Control, Comput.*, Sept. 2005.

[2] D. Baron, M. B. Wakin, M. F. Duarte, S. Sarvotham, and R. G. Baraniuk. Distributed compressed sensing. 2005. Preprint. Available at www.dsp.rice.edu/cs.

[3] E. Candès, J. Romberg, and T. Tao. Stable signal recovery from incomplete and inaccurate measurements. *Comm. Pure Applied Mathematics*, 2005. To appear.

[4] E. Candès and T. Tao. Near optimal signal recovery from random projections and universal encoding strategies. 2004. Preprint.

[5] E. Candès and T. Tao. The Dantzig selector: Statistical estimation when $p$ is much larger than $n$. 2005. Preprint.

[6] E. Candès and T. Tao. Error correction via linear programming. 2005. Preprint.

[7] S. Chen, D. Donoho, and M. Saunders. Atomic decomposition by basis pursuit. *SIAM Journal on Scientific Computing*, 20(1):33–61, 1998.

[8] T. M. Cover and J. A. Thomas. *Elements of Information Theory*. Wiley, New York, 1991.

[9] D. Donoho. Compressed sensing. 2004. Preprint.

[10] D. Donoho and Y. Tsaig. Extensions of compressed sensing. 2004. Preprint.

[11] M. F. Duarte, S. Sarvotham, D. Baron, M. B. Wakin, and R. G. Baraniuk. Distributed compressed sensing of jointly sparse signals. In *Asilomar Conf. Signals, Sys., Comput.*, Nov. 2005.

[12] J. Haupt and R. Nowak. Signal reconstruction from noisy random projections. 2005. Preprint.

[13] S. Pradhan and K. Ramchandran. Distributed source coding using syndromes (DISCUS): Design and construction. *IEEE Trans. Inform. Theory*, 49:626–643, March 2003.

[14] D. Slepian and J. K. Wolf. Noiseless coding of correlated information sources. *IEEE Trans. Inform. Theory*, 19:471–480, July 1973.

[15] J. Tropp and A. C. Gilbert. Signal recovery from partial information via orthogonal matching pursuit. 2005. Preprint.

[16] J. Tropp, A. C. Gilbert, and M. J. Strauss. Simulataneous sparse approximation via greedy pursuit. In *IEEE 2005 Int. Conf. Acoustics, Speech, Signal Processing*, March 2005.

[17] Z. Xiong, A. Liveris, and S. Cheng. Distributed source coding for sensor networks. *IEEE Signal Proc. Mag.*, 21:80–94, September 2004.